# Exploiting Tractable Substructures in Intractable Networks

**Lawrence K. Saul and Michael I. Jordan**
{lksaul,jordan}@psyche.mit.edu

Center for Biological and Computational Learning
Massachusetts Institute of Technology
79 Amherst Street, E10-243
Cambridge, MA 02139

## Abstract

We develop a refined mean field approximation for inference and learning in probabilistic neural networks. Our mean field theory, unlike most, does not assume that the units behave as independent degrees of freedom; instead, it exploits in a principled way the existence of large substructures that are computationally tractable. To illustrate the advantages of this framework, we show how to incorporate weak higher order interactions into a first-order hidden Markov model, treating the corrections (but not the first order structure) within mean field theory.

## 1 INTRODUCTION

Learning the parameters in a probabilistic neural network may be viewed as a problem in statistical estimation. In networks with sparse connectivity (e.g. trees and chains), there exist efficient algorithms for the exact probabilistic calculations that support inference and learning. In general, however, these calculations are intractable, and approximations are required.

Mean field theory provides a framework for approximation in probabilistic neural networks (Peterson & Anderson, 1987). Most applications of mean field theory, however, have made a rather drastic probabilistic assumption—namely, that the units in the network behave as independent degrees of freedom. In this paper we show how to go beyond this assumption. We describe a self-consistent approximation in which tractable substructures are handled by exact computations and only the remaining, intractable parts of the network are handled within mean field theory. For simplicity we focus on networks with binary units; the extension to discrete-valued (Potts) units is straightforward.

We apply these ideas to hidden Markov modeling (Rabiner & Juang, 1991). The first order probabilistic structure of hidden Markov models (HMMs) leads to networks with chained architectures for which efficient, exact algorithms are available. More elaborate networks are obtained by introducing couplings between multiple HMMs (Williams & Hinton, 1990) and/or long-range couplings within a single HMM (Stolorz, 1994). Both sorts of extensions have interesting applications; in speech, for example, multiple HMMs can provide a distributed representation of the articulatory state, while long-range couplings can model the effects of coarticulation. In general, however, such extensions lead to networks for which exact probabilistic calculations are not feasible. One would like to develop a mean field approximation for these networks that exploits the tractability of first-order HMMs. This is possible within the more sophisticated mean field theory described here.

## 2 MEAN FIELD THEORY

We briefly review the basic methodology of mean field theory for networks of binary ($\pm 1$) stochastic units (Parisi, 1988). For each configuration $\{S\} = \{S_1, S_2, \ldots, S_N\}$, we define an energy $E\{S\}$ and a probability $P\{S\}$ via the Boltzmann distribution:

$$P\{S\} = \frac{e^{-\beta E\{S\}}}{Z}, \tag{1}$$

where $\beta$ is the inverse temperature and $Z$ is the partition function. When it is intractable to compute averages over $P\{S\}$, we are motivated to look for an approximating distribution $Q\{S\}$. Mean field theory posits a particular parametrized form for $Q\{S\}$, then chooses parameters to minimize the Kullback-Liebler (KL) divergence:

$$\text{KL}(Q\|P) = \sum_{\{S\}} Q\{S\} \ln \left[ \frac{Q\{S\}}{P\{S\}} \right]. \tag{2}$$

Why are mean field approximations valuable for learning? Suppose that $P\{S\}$ represents the posterior distribution over hidden variables, as in the E-step of an EM algorithm (Dempster, Laird, & Rubin, 1977). Then we obtain a mean field approximation to this E-step by replacing the statistics of $P\{S\}$ (which may be quite difficult to compute) with those of $Q\{S\}$ (which may be much simpler). If, in addition, $Z$ represents the likelihood of observed data (as is the case for the example of section 3), then the mean field approximation yields a lower bound on the log-likelihood. This can be seen by noting that for any approximating distribution $Q\{S\}$, we can form the lower bound:

$$\ln Z = \ln \sum_{\{S\}} e^{-\beta E\{S\}} \tag{3}$$

$$= \ln \sum_{\{S\}} Q\{S\} \cdot \left[ \frac{e^{-\beta E\{S\}}}{Q\{S\}} \right] \tag{4}$$

$$\geq \sum_{\{S\}} Q\{S\} \left[ -\beta E\{S\} - \ln Q\{S\} \right], \tag{5}$$

where the last line follows from Jensen's inequality. The difference between the left and right-hand side of eq. (5) is exactly $\text{KL}(Q\|P)$; thus the better the approximation to $P\{S\}$, the tighter the bound on $\ln Z$. Once a lower bound is available, a learning procedure can maximize the lower bound. This is useful when the true likelihood itself cannot be efficiently computed.

## 2.1 Complete Factorizability

The simplest mean field theory involves assuming marginal independence for the units $S_i$. Consider, for example, a quadratic energy function

$$-\beta E\{S\} = \sum_{i<j} J_{ij} S_i S_j + \sum_i h_i S_i, \qquad (6)$$

and the factorized approximation:

$$Q\{S\} = \prod_i \left( \frac{1 + m_i S_i}{2} \right). \qquad (7)$$

The expectations under this mean field approximation are $\langle S_i \rangle = m_i$ and $\langle S_i S_j \rangle = m_i m_j$ for $i \neq j$. The best approximation of this form is found by minimizing the KL-divergence,

$$\begin{aligned} \mathrm{KL}(Q\|P) &= \sum_i \left[ \left( \frac{1+m_i}{2} \right) \ln \left( \frac{1+m_i}{2} \right) + \left( \frac{1-m_i}{2} \right) \ln \left( \frac{1-m_i}{2} \right) \right] \quad (8) \\ &\quad - \sum_{i<j} J_{ij} m_i m_j - \sum_i h_i m_i + \ln Z, \end{aligned}$$

with respect to the mean field parameters $m_i$. Setting the gradients of eq. (8) equal to zero, we obtain the (classical) mean field equations:

$$\tanh^{-1}(m_i) = \sum_j J_{ij} m_j + h_i. \qquad (9)$$

## 2.2 Partial Factorizability

We now consider a more structured model in which the network consists of interacting modules that, taken in isolation, define tractable substructures. One example of this would be a network of weakly coupled HMMs, in which each HMM, taken by itself, defines a chain-like substructure that supports efficient probabilistic calculations. We denote the interactions between these modules by parameters $K_{ij}^{\mu\nu}$, where the superscripts $\mu$ and $\nu$ range over modules and the subscripts $i$ and $j$ index units within modules. An appropriate energy function for this network is:

$$-\beta E\{S\} = \sum_\mu \left\{ \sum_{i<j} J_{ij}^\mu S_i^\mu S_j^\mu + \sum_i h_i^\mu S_i^\mu \right\} + \sum_{\substack{\mu<\nu \\ ij}} K_{ij}^{\mu\nu} S_i^\mu S_j^\nu. \qquad (10)$$

The first term in this energy function contains the intra-modular interactions; the last term, the inter-modular ones.

We now consider a mean field approximation that maintains the first sum over modules but dispenses with the inter-modular corrections:

$$Q\{S\} = \frac{1}{Z_Q} \exp \left\{ \sum_\mu \left[ \sum_{i<j} J_{ij}^\mu S_i^\mu S_j^\mu + \sum_i H_i^\mu S_i^\mu \right] \right\} \qquad (11)$$

The parameters of this mean field approximation are $H_i^\mu$; they will be chosen to provide a self-consistent model of the inter-modular interactions. We easily obtain the following expectations under the mean field approximation, where $\mu \neq \nu$:

$$\langle S_i^\mu S_j^\omega \rangle = \delta_{\mu\omega} \langle S_i^\omega S_j^\omega \rangle + (1 - \delta_{\mu\omega}) \langle S_i^\mu \rangle \langle S_j^\omega \rangle, \qquad (12)$$

$$\begin{aligned} \langle S_i^\mu S_j^\nu S_k^\omega \rangle &= \delta_{\mu\omega} \langle S_i^\omega S_k^\omega \rangle \langle S_j^\nu \rangle + \delta_{\nu\omega} \langle S_j^\omega S_k^\omega \rangle \langle S_i^\mu \rangle + \qquad (13) \\ &\quad (1 - \delta_{\nu\omega})(1 - \delta_{\omega\mu}) \langle S_i^\mu \rangle \langle S_j^\nu \rangle \langle S_k^\omega \rangle. \end{aligned}$$

Note that units in the same module are statistically correlated and that these correlations are assumed to be taken into account in calculating the expectations. We assume that an efficient algorithm is available for handling these intra-modular correlations. For example, if the factorized modules are chains (e.g. obtained from a coupled set of HMMs), then computing these expectations requires a forward-backward pass through each chain.

The best approximation of the form, eq. (11), is found by minimizing the KL-divergence,

$$\mathrm{KL}(Q||P) = \ln(Z/Z_Q) + \sum_{\mu i}(H_i^\mu - h_i^\mu)\langle S_i^\mu\rangle - \sum_{\substack{\mu < \nu \\ ij}} K_{ij}^{\mu\nu}\langle S_i^\mu S_j^\nu\rangle, \qquad (14)$$

with respect to the mean field parameters $H_k^\omega$. To compute the appropriate gradients, we use the fact that derivatives of expectations under a Boltzmann distribution (e.g. $\partial\langle S_i^\mu\rangle/\partial H_k^\omega$) yield cumulants (e.g. $\langle S_i^\mu S_k^\omega\rangle - \langle S_i^\mu\rangle\langle S_k^\omega\rangle$). The conditions for stationarity are then:

$$0 = \sum_{\mu i}(H_i^\mu - h_i^\mu)[\langle S_i^\mu S_k^\omega\rangle - \langle S_i^\mu\rangle\langle S_k^\omega\rangle] - \sum_{\substack{\mu < \nu \\ ij}} K_{ij}^{\mu\nu}\left[\langle S_i^\mu S_j^\nu S_k^\omega\rangle - \langle S_i^\mu S_j^\nu\rangle\langle S_k^\omega\rangle\right]. \quad (15)$$

Substituting the expectations from eqs. (12) and (13), we find that $KL(Q||P)$ is minimized when

$$0 = \sum_i\left\{H_i^\omega - h_i^\omega - \sum_{\nu\neq\omega}\sum_j K_{ij}^{\omega\nu}\langle S_j^\nu\rangle\right\}[\langle S_i^\omega S_k^\omega\rangle - \langle S_i^\omega\rangle\langle S_k^\omega\rangle]. \qquad (16)$$

The resulting mean field equations are:

$$H_i^\omega = \sum_{\nu\neq\omega}\sum_j K_{ij}^{\omega\nu}\langle S_j^\nu\rangle + h_i^\omega. \qquad (17)$$

These equations may be solved by iteration, in which the (assumed) tractable algorithms for averaging over $Q\{S\}$ are invoked as subroutines to compute the expectations $\langle S_j^\nu\rangle$ on the right hand side. Because these expectations depend on $H_i^\nu$, these equations may be viewed as a self-consistent model of the inter-modular interactions. Note that the mean field parameter $H_i^\omega$ plays a role analogous to $\tanh^{-1}(m_i)$ in eq. (9) of the fully factorized case.

## 2.3  Inducing Partial Factorizability

Many interesting networks do not have strictly modular architectures and can only be approximately decomposed into tractable core structures. Techniques are needed in such cases to induce partial factorizability. Suppose for example that we are given an energy function

$$-\beta E\{S\} = \sum_{i<j} J_{ij}S_iS_j + \sum_i h_iS_i + \sum_{i<j} K_{ij}S_iS_j \qquad (18)$$

for which the first two terms represent tractable interactions and the last term, intractable ones. Thus the weights $J_{ij}$ by themselves define a tractable skeleton network, but the weights $K_{ij}$ spoil this tractability. Mimicking the steps of the previous section, we obtain the mean field equations:

$$0 = \sum_i(\langle S_iS_k\rangle - \langle S_i\rangle\langle S_k\rangle)[H_i - h_i] - \sum_{i<j} K_{ij}[\langle S_iS_jS_k\rangle - \langle S_iS_j\rangle\langle S_k\rangle]. \quad (19)$$

In this case, however, the weights $K_{ij}$ couple units in the same core structure. Because these units are not assumed to be independent, the triple correlator $\langle S_i S_j S_k \rangle$ does not factorize, and we no longer obtain the decoupled update rules of eq. (17). Rather, for these mean field equations, each iteration requires computing triple correlators and solving a large set of coupled linear equations.

To avoid this heavy computational load, we instead manipulate the energy function into one that can be partially factorized. This is done by introducing extra hidden variables $W_{ij} = \pm 1$ on the intractable links of the network. In particular, consider the energy function

$$-\beta E\{S,W\} = \sum_{i<j} J_{ij} S_i S_j + \sum_i h_i S_i + \sum_{i<j} \left[ K_{ij}^{(1)} S_i + K_{ij}^{(2)} S_j \right] W_{ij}. \quad (20)$$

The hidden variables $W_{ij}$ in eq. (20) serve to decouple the units connected by the intractable weights $K_{ij}$. However, we can always choose the new interactions, $K_{ij}^{(1)}$ and $K_{ij}^{(2)}$, so that

$$e^{-\beta E\{S\}} = \sum_{\{W\}} e^{-\beta E\{S,W\}}. \quad (21)$$

Eq. (21) states that the marginal distribution over $\{S\}$ in the new network is identical to the joint distribution over $\{S\}$ in the original one. Summing both sides of eq. (21) over $\{S\}$, it follows that both networks have the same partition function.

The form of the energy function in eq. (20) suggests the mean field approximation:

$$Q\{S,W\} = \frac{1}{Z_Q} \exp \left\{ \sum_{i<j} J_{ij} S_i S_j + \sum_i H_i S_i + \sum_{i<j} H_{ij} W_{ij} \right\}, \quad (22)$$

where the mean field parameters $H_i$ have been augmented by a set of additional mean field parameters $H_{ij}$ that account for the extra hidden variables. In this expression, the variables $S_i$ and $W_{ij}$ act as decoupled degrees of freedom and the methods of the preceding section can be applied directly. We consider an example of this reduction in the following section.

## 3   EXAMPLE

Consider a continuous-output HMM in which the probability of an output $\vec{X}_t$ at time $t$ is dependent not only on the state at time $t$, but also on the state at time $t + \Delta$. Such a context-sensitive HMM may serve as a flexible model of anticipatory coarticulatory effects in speech, with $\Delta \approx 50$ms representing a mean phoneme lifetime. Incorporating these interactions into the basic HMM probability model, we obtain the following joint probability on states and outputs:

$$P\{S,\vec{X}\} = \prod_{t=1}^{T-1} a_{S_t S_{t+1}} \prod_{t=1}^{T-\Delta} \frac{1}{(2\pi)^{D/2}} \exp \left\{ -\frac{1}{2} \left[ \vec{X}_t - \vec{U}_{S_t} - \vec{V}_{S_{t+\Delta}} \right]^2 \right\}. \quad (23)$$

Denoting the likelihood of an output sequence by $Z$, we have

$$Z = P\{\vec{X}\} = \sum_{\{S\}} P\{S,\vec{X}\}. \quad (24)$$

We can represent this probability model using energies rather than transition probabilities (Luttrell, 1989; Saul and Jordan, 1995). For the special case of binary

states, this is done by choosing weights $J$, $K$, and $h_t$ related to the parameters of the HMM and the output sequence as follows[1]:

$$J \;=\; \frac{1}{4}\ln\left[\frac{a_{++}a_{--}}{a_{+-}a_{-+}}\right], \qquad K = -\frac{1}{4}(\vec{U}_+ - \vec{U}_-)\cdot(\vec{V}_+ - \vec{V}_-), \qquad (25)$$

$$h_t \;=\; \frac{1}{2}\ln\left[\frac{a_{++}}{a_{--}}\right] + \frac{1}{2}\left[\vec{X}_t - \frac{\vec{U}_+ + \vec{U}_- + \vec{V}_+ + \vec{V}_-}{2}\right]\cdot\left[\vec{U}_+ + \vec{V}_+ - \vec{U}_- - \vec{V}_-\right] (26)$$

Here, $a_{++}$ is the probability of transitioning from the ON state to the ON state (and similarly for the other $a$ parameters), while $\vec{U}_+$ and $\vec{V}_+$ are the mean outputs associated with the ON state at time steps $t$ and $t + \Delta$ (and similarly for $\vec{U}_-$ and $\vec{V}_-$). Given these definitions, we obtain an equivalent expression for the likelihood:

$$Z = \sum_{\{S\}} \exp\left\{-\varepsilon_0 + \sum_{t=1}^{T-1} J S_t S_{t+1} + \sum_{t=1}^{T} h_t S_t + \sum_{t=1}^{T-\Delta} K S_t S_{t+\Delta}\right\}, \qquad (27)$$

where $\varepsilon_0$ is a placeholder for the terms in $\ln P\{S, \vec{X}\}$ that do not depend on $\{S\}$. We can interpret $Z$ as the partition function for the chained network of $T$ binary units that represents the HMM unfolded in time. The nearest neighbor connectivity of this network reflects the first order structure of the HMM; the long-range connectivity reflects the higher order interactions that model sensitivity to context.

The exact likelihood can in principle be computed by summing over the hidden states in eq. (27), but the required forward-backward algorithm scales much worse than the case of first-order HMMs. Because the likelihood can be identified as a partition function, however, we can obtain a lower bound on its value from mean field theory. To exploit the tractable first order structure of the HMM, we induce a partially factorizable network by introducing extra link variables on the long-range connections, as described in section 2.3. The resulting mean field approximation uses the chained structure as its backbone and should be accurate if the higher order effects in the data are weak compared to the basic first-order structure.

The above scenario was tested in numerical simulations. In actuality, we implemented a generalization of the model in eq. (23): our HMM had non-binary hidden states and a coarticulation model that incorporated both left and right context. This network was trained on several artificial data sets according to the following procedure. First, we fixed the "context" weights to zero and used the Baum-Welch algorithm to estimate the first order structure of the HMM. Then, we lifted the zero constraints and re-estimated the parameters of the HMM by a mean field EM algorithm. In the E-step of this algorithm, the true posterior $P\{S|\vec{X}\}$ was approximated by the distribution $Q\{S|\vec{X}\}$ obtained by solving the mean field equations; in the M-step, the parameters of the HMM were updated to match the statistics of $Q\{S|\vec{X}\}$. Figure 1 shows the type of structure captured by a typical network.

# 4    CONCLUSIONS

Endowing networks with probabilistic semantics provides a unified framework for incorporating prior knowledge, handling missing data, and performing inferences under uncertainty. Probabilistic calculations, however, can quickly become intractable, so it is important to develop techniques that both approximate probability distributions in a flexible manner and make use of exact techniques wherever possible. In

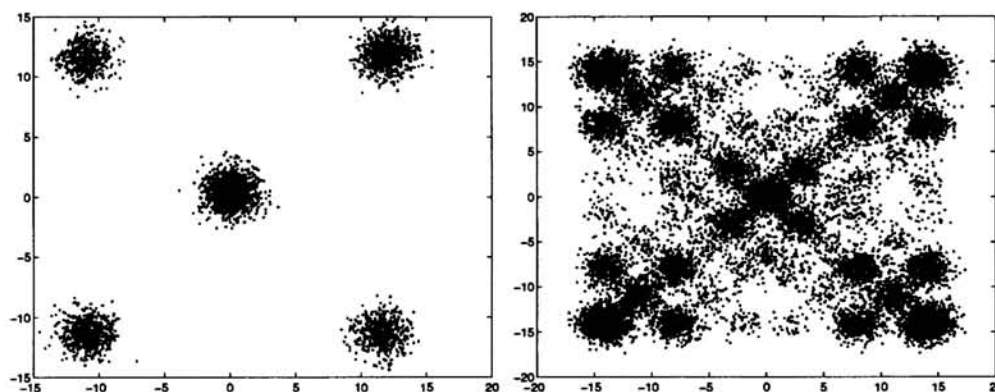

Figure 1: 2D output vectors $\{\vec{X}_t\}$ sampled from a first-order HMM and a context-sensitive HMM, each with $n = 5$ hidden states. The latter's coarticulation model used left and right context, coupling $\vec{X}_t$ to the hidden states at times $t$ and $t \pm 5$. At left: the five main clusters reveal the basic first-order structure. At right: weak modulations reveal the effects of context.

this paper we have developed a mean field approximation that meets both these objectives. As an example, we have applied our methods to context-sensitive HMMs, but the methods are general and can be applied more widely.

## Acknowledgements

The authors acknowledge support from NSF grant CDA-9404932, ONR grant N00014-94-1-0777, ATR Research Laboratories, and Siemens Corporation.

## Footnotes

[1]There are boundary corrections to $h_t$ (not shown) for $t = 1$ and $t > T - \Delta$.

## References

A. Dempster, N. Laird, and D. Rubin. (1977) Maximum likelihood from incomplete data via the EM algorithm. *J. Roy. Stat. Soc.* **B39**:1-38.

B. H. Juang and L. R. Rabiner. (1991) Hidden Markov models for speech recognition, *Technometrics* **33**: 251-272.

S. Luttrell. (1989) The Gibbs machine applied to hidden Markov model problems. *Royal Signals and Radar Establishment: SP Research Note* **99**.

G. Parisi. (1988) *Statistical field theory.* Addison-Wesley: Redwood City, CA.

C. Peterson and J. R. Anderson. (1987) A mean field theory learning algorithm for neural networks. *Complex Systems* **1**:995-1019.

L. Saul and M. Jordan. (1994) Learning in Boltzmann trees. *Neural Comp.* **6**: 1174-1184.

L. Saul and M. Jordan. (1995) Boltzmann chains and hidden Markov models. In G. Tesauro, D. Touretzky, and T. Leen, eds. *Advances in Neural Information Processing Systems* **7**. MIT Press: Cambridge, MA.

P. Stolorz. (1994) Recursive approaches to the statistical physics of lattice proteins. In L. Hunter, ed. *Proc. 27th Hawaii Intl. Conf. on System Sciences* **V**: 316-325.

C. Williams and G. E. Hinton. (1990) Mean field networks that learn to discriminate temporally distorted strings. *Proc. Connectionist Models Summer School*: 18-22.